# Using Bayesian Dynamical Systems for Motion Template Libraries

**Silvia Chiappa,  Jens Kober,  Jan Peters**

Max-Planck Institute for Biological Cybernetics
Spemannstraße 38, 72076 Tübingen, Germany
`{silvia.chiappa,jens.kober,jan.peters}@tuebingen.mpg.de`

## Abstract

Motor primitives or motion templates have become an important concept for both modeling human motor control as well as generating robot behaviors using imitation learning. Recent impressive results range from humanoid robot movement generation to timing models of human motions. The automatic generation of skill libraries containing multiple motion templates is an important step in robot learning. Such a skill learning system needs to cluster similar movements together and represent each resulting motion template as a generative model which is subsequently used for the execution of the behavior by a robot system. In this paper, we show how human trajectories captured as multi-dimensional time-series can be clustered using Bayesian mixtures of linear Gaussian state-space models based on the similarity of their dynamics. The appropriate number of templates is automatically determined by enforcing a parsimonious parametrization. As the resulting model is intractable, we introduce a novel approximation method based on variational Bayes, which is especially designed to enable the use of efficient inference algorithms. On recorded human Balero movements, this method is not only capable of finding reasonable motion templates but also yields a generative model which works well in the execution of this complex task on a simulated anthropomorphic SARCOS arm.

## 1   Introduction

Humans demonstrate a variety and versatility of movements far beyond the reach of current anthropomorphic robots. It is widely believed that human motor control largely relies on a set of "mental templates" [1] better known as *motor primitives* or *motion templates*. This concept has gained increasing attention both in the human motor control literature [1, 2] as well as in robot imitation learning [3, 4]. The recent suggestion of Ijspeert et al. [3] to use dynamical systems as motor primitives has allowed this approach to scale in the domain of humanoid robot imitation learning and has yielded a variety of interesting applications as well as follow-up publications. However, up to now, the focus of motion template learning has largely been on single template acquisition and self-improvement. Future motor skill learning systems on the other hand need to be able to observe several different behaviors from human presenters and compile libraries of motion templates directly from these examples with as little predetermined structures as possible.

An important part of such a motor skill learning system is the clustering of many presented movements into different motion templates. Human trajectories are recorded as multi-dimensional time-series of joint angles as well as joint velocities using either a marker-based tracking setup (e.g., a VICON$^{TM}$ setup), a sensing suit (e.g., a SARCOS SenSuit) or a haptic interface (e.g., an anthropomorphic master arm). Inspired by Ijspeert et al. [3], we intend to use dynamical systems as generative models of the presented trajectories, i.e., as motion templates. Our goal is to cluster

these multi-dimensional time-series automatically into a small number of motion templates without pre-labeling of the trajectories or assuming an a priori number of templates. Thus, the system has to discover the underlying motion templates, determine the number of templates as well as learn the underlying skill sufficiently well for robot application.

In principle, one could use a non-generative clustering approach (e.g., a type of K-means) with a method for selecting an appropriate number of clusters and, subsequently, fit a generative model to each cluster. Here we prefer to take a different approach in which the clustering and learning of the underlying time-series dynamics are performed at the same time. This way we aim at ensuring that each obtained cluster can be modeled well by its representative generative model.

To date the majority of the work on time-series clustering using generative models has focused on static mixture models. Clustering long or high-dimensional time-series is hard when approached with static models, such that collapsing the trajectories to a few relevant features is often required. This problem would be severe for a high-dimensional motor learning system where the data needs to be represented at high sampling rates in order to ensure the capturing of all relevant details for motor skill learning. In addition, it is difficult to ensure smoothness when the time-series display high variability and, therefore, to obtain accurate generative models with static approaches.

A natural alternative is to use mixtures of temporal models which explicitly model the dynamics of the time-series. In this paper, we use Mixtures of Linear Gaussian State-Space Models (LGSSMs). LGSSMs are probabilistic temporal models which, despite their computational simplicity, can represent many natural dynamical processes [5]. As we will see later in this paper, LGSSMs are powerful enough to model our time-series sufficiently accurately.

For determining the number of clusters, most probabilistic approaches in the past used to train a separate model for each possible cluster configuration, and then select the one which would optimize the trade-off between accuracy and complexity, as measured for example by the Bayesian Information Criterion [6, 7]. The drawback of these approaches is that training many separate models can lead to a large computational overhead, such that heuristics are often needed to restrict the number of possible cluster configurations [7].

A less computationally expensive alternative is offered by recent Bayesian approaches where the model parameters are treated as random variables and integrated out yielding the marginal likelihood of the data. An appropriate prior distribution can be used to enforce a sparse representation, i.e., to select the smallest set of parameters that explains the data well by making the remaining parameters inactive. As a result, the structure selection can be achieved within the model, without the need to train and compare several separate models.

As a Bayesian treatment of the Mixtures of Linear Gaussian State-Space Models is intractable, we introduce a deterministic approximation based on variational Bayes. Importantly, our approximation is especially designed to enable the use of standard LGSSM inference methods for the hidden state variables, which has the advantage of minimizing numerically instabilities.

As a realistically difficult scenario in this first step towards large motor skill libraries, we have selected the game of dexterity Balero (also known as Ball-In-A-Cup or Kendama, see [8]) as an evaluation platform. Several substantially different types of movements exist for performing this task and humans tend to have a large variability in movement execution [9]. From a robotics point of view, Balero can be considered sufficiently complex as it involves movements in all major seven degrees of freedom of a human arm as well as an anthropomorphic robot arm. We are able to show that the presented method gives rise to a reasonable number of clusters representing quite distinct movements and that the resulting generative models can be used successfully as motion templates in physically realistic simulations.

In the remainder of the paper, we will proceed as follows. We will first introduce a generative approach for clustering and modeling multi-dimensional time-series with Bayesian Mixtures of LGSSMs and describe how this approach can be made tractable using a variational approximation. We will then show that the resulting model can be used to infer the motion templates underlying a set of human demonstrations, and give evidence that the generative model representing each motion template is sufficiently accurate for control in a mechanically plausible simulation of the SARCOS Master Arm.

## 2 Bayesian Mixtures of Linear Gaussian State-Space Models

Our goal is to model both human and robot movements in order to build motion template libraries. In this section, we describe our Bayesian modeling approach and discuss both the underlying assumptions as well as how the structure of the model is selected. As the resulting model is not tractable for analytical solution, we introduce an approximation method based on variational Bayes.

### 2.1 Modeling Approach

In our Bayesian approach to Mixtures of Linear Gaussian State-Space Models (LGSSMs), we are given a set of $N$ time-series[1] $v_{1:T}^{1:N}$ of length $T$ for which we define with the following marginal likelihood

$$p(v_{1:T}^{1:N}|\hat{\Theta}^{1:K}, \gamma) = \sum_{z^{1:N}} \int_{\Theta^{1:K}} p(v_{1:T}^{1:N}|z^{1:N}, \Theta^{1:K}) p(\Theta^{1:K}|\hat{\Theta}^{1:K}) \int_{\pi} p(z^{1:N}|\pi)p(\pi|\gamma),$$

where $z^n \in \{1, \ldots, K\}$ indicates which of a set of $K$ LGSSMs generated the sequence $v_{1:T}^n$. The parameters of LGSSM $k$ are denoted by $\Theta^k$ and have a prior distribution depending on hyperparameters $\hat{\Theta}^k$. The $K$-dimensional vector $\pi$ includes the prior probabilities of the time-series generation for each LGSSM and has prior distribution hyperparameter $\gamma$.

The optimal hyperparameters are estimated by type-II maximum likelihood [10], i.e., by maximizing the marginal likelihood over $\hat{\Theta}^{1:K}$ and $\gamma$. Clustering can be performed by inferring the LGSSM that most likely generated the sequence $v_{1:T}^n$ by computing $\arg\max_k p(z^n = k|v_{1:T}^{1:N}, \hat{\Theta}^{1:K}, \gamma)$.

**Modeling $p(v_{1:T}^{1:N}|z^{1:N}, \hat{\Theta}^{1:K})$.** As a generative temporal model for each time-series, we employ a Linear Gaussian State-Space Model [5] that assumes that the observations $v_{1:T}$, with $v_t \in \Re^V$, are generated from a latent Markovian linear dynamical system with hidden states $h_{1:T}$, with $h_t \in \Re^H$, according to[2]

$$v_t = Bh_t + \eta_t^v, \eta_t^v \sim \mathcal{N}(0_V, \Sigma_V), \qquad h_t = Ah_{t-1} + \eta_t^h, \eta_t^h \sim \mathcal{N}(\mu_t, \Sigma_H). \tag{1}$$

Standard LGSSMs assume a zero-mean hidden-state noise ($\mu_t \equiv 0_H$). In our application the use of a time-dependent mean $\mu_t \neq 0_H$ leads to a superior modeling accuracy. A probabilistic formulation of the LGSSM is given by

$$p(v_{1:T}, h_{1:T}|\Theta) = p(v_1|h_1, \Theta)p(h_1|\Theta) \prod_{t=2}^{T} p(v_t|h_t, \Theta)p(h_t|h_{t-1}, \Theta),$$

with $p(h_t|h_{t-1}, \Theta) = \mathcal{N}(Ah_{t-1} + \mu_t, \Sigma_H)$, $p(h_1|\Theta) = \mathcal{N}(\mu_1, \Sigma)$, $p(v_t|h_t, \Theta) = \mathcal{N}(Bh_t, \Sigma_V)$, and $\Theta = \{A, B, \Sigma_H, \Sigma_V, \mu_{1:T}, \Sigma\}$. Due to the simple structure of the model, performing inference, that is to compute quantities such as $p(h_t|v_{1:T}, \Theta)$, can be efficiently achieved in $O(T)$ operations.

In the presented Bayesian approach, we define a prior distribution $p(\Theta|\hat{\Theta})$ over the parameters $\Theta$ where $\hat{\Theta}$ are the associated hyperparameters. More specifically, we define zero-mean Gaussians on the elements of $A$ and on the columns of $B$ by[3]

$$p\left(A|\alpha, \Sigma_H^{-1}\right) = \prod_{i,j=1}^{H} \frac{\alpha_{ij}^{1/2}}{\sqrt{2\pi[\Sigma_H]_{ii}}} e^{-\frac{\alpha_{ij}}{2}[\Sigma_H^{-1}]_{ii} A_{ij}^2}, p\left(B|\beta, \Sigma_V^{-1}\right) = \prod_{j=1}^{H} \frac{\beta_j^{V/2}}{\sqrt{|2\pi\Sigma_V|}} e^{-\frac{\beta_j}{2} B_j^\mathsf{T} \Sigma_V^{-1} B_j},$$

where $\alpha$ and $\beta$ are a set of hyperparameters which need to be optimized. We make the assumption that $\Sigma_H^{-1}$, $\Sigma_V^{-1}$ and $\Sigma^{-1}$ are diagonal and define Gamma distributions on them. For $\mu_1$ we define a zero-mean Gaussian prior, while we formally treat $\mu_{2:T}$ as hyperparameters and determine their

optimal values. These choices are made in order to render our Bayesian treatment feasible and to obtain a sparse parametrization, as discussed in more details below.

In the resulting mixture model, we consider a set of $K$ such Bayesian LGSSMs. The joint distribution over all sequences given the indicator variables and hyperparameters is defined as

$$p(v_{1:T}^{1:N}|z^{1:N}, \hat{\Theta}^{1:K}) = \int_{\Theta^{1:K}} \left\{ \prod_{n=1}^{N} p(v_{1:T}^n|z^n, \Theta^{1:K}) \right\} \prod_{k=1}^{K} p(\Theta^k|\hat{\Theta}^k),$$

where $p(v_{1:T}^n|z^n = k, \Theta^{1:K}) \equiv p(v_{1:T}^n|\Theta^k)$ denotes the probability of time-series $v_{1:T}^n$ given that parameters $\Theta^k$ have been employed to generate it.

**Modeling $p(z^{1:N}|\gamma)$.** As prior for $\pi$, we define a symmetric Dirichlet distribution

$$p(\pi|\gamma) = \frac{\Gamma(\gamma)}{\Gamma(\gamma/K)^K} \prod_{k=1}^{K} \pi_k^{\gamma/K-1},$$

where $\Gamma(\cdot)$ is the Gamma function and $\gamma$ denotes a hyperparameter that needs to be optimized. This distribution is conjugate to the multinomial, which greatly simplifies our Bayesian treatment. To model the joint indicator variables, we define

$$p(z^{1:N}|\gamma) = \int_{\pi} \left\{ \prod_{n=1}^{N} p(z^n|\pi) \right\} p(\pi|\gamma), \quad \text{where } p(z^n = k|\pi) \equiv \pi_k.$$

Such Bayesian approach favors simple model structures. In particular, the priors on $A^k$ and $B^k$ enforce a sparse parametrization since, during learning, many $\alpha_{ij}^k$ and $\beta_j^k$ get close to infinity whereby (the posterior distribution of) $A_{ij}^k$ and $B_j^k$ get close to zero (see [11] for an analysis of this pruning effect). This enables us to achieve structure selection within the model. Specifically, this approach ensures that the unnecessary LGSSMs are pruned out from the model during training (for certain $k$, all elements of $B^k$ are pruned out such that LGSSM $k$ becomes inactive ($p(z^n = k|v_{1:T}^{1:N}, \hat{\Theta}^{1:K}, \gamma) = 0$ for all $n$)).

## 2.2 Model Intractability and Approximate Solution

The Bayesian treatment of the model is non-trivial as the integration over the parameters $\Theta^{1:K}$ and $\pi$ renders the computation of the required posterior distributions intractable. This problem results from the coupling in the posterior distributions between the hidden state variables $h_{1:T}^{1:N}$ and the parameters $\Theta^{1:K}$ as well as between the indicators $z^{1:N}$ and $\pi, \Theta^{1:K}$. To deal with this intractability, we use a deterministic approximation method based on variational Bayes.

**Variational Approximation.** In our variational approach we introduce a new distribution $q$ and make the following approximation[4]

$$p(z^{1:N}, h_{1:T}^{1:N}, \Theta^{1:K}|v_{1:T}^{1:N}, \hat{\Theta}^{1:K}, \gamma) \approx q(h_{1:T}^{1:N}|z^{1:N})q(z^{1:N})q(\Theta^{1:K}). \quad (2)$$

That is, we approximate the posterior distribution of the hidden variables of the model by one in which the hidden states are decoupled from the parameters given the indicator variables and in which the indicators are decoupled from the parameters.

The approximation is achieved with a variational expectation-maximization algorithm which minimizes the KL divergence between the right and left hand sides of Equation (2), or, equivalently, maximizes a tractable lower bound on the log-likelihood $\log p(v_{1:T}^{1:N}|\hat{\Theta}^{1:K}, \gamma) \geq \mathcal{F}(\hat{\Theta}^{1:K}, \gamma, q)$ with respect to $q$ for fixed $\hat{\Theta}^{1:K}$ and $\gamma$ and vice-versa. Observation $v_t^n$ is then placed in the most likely LGSSM by computing $\arg\max_k q(z^n = k)$.

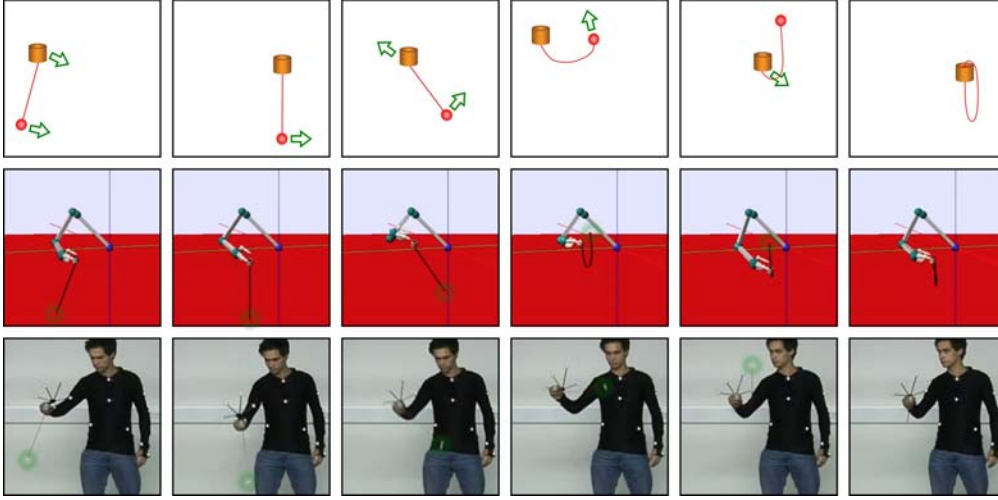

Figure 1: This figure shows one of the Balero motion templates found by our clustering method, i.e., the cluster C2 in Figure 2. Here, a sideways movement with a subsequent catch is performed and the uppermost row illustrates this movement with a symbolic sketch. The middle row shows an execution of the movement generated with the LGSSM representing the cluster C2. The lowest row shows a recorded human movement which was attributed to cluster C2 by our method. Note that movements generated from LGSSMs representing other clusters differ significantly.

**Resulting Updates.** While the space does not suffice for complete derivation, we will briefly sketch the updates for $q$. Additional details and the updates for the hyperparameters can be found in [12]. The updates consist of a parameter update, an indicator variable update and a latent state update. First, the approximate parameter posterior is given by

$$q(\Theta^k) \propto p(\Theta^k|\hat{\Theta}^k)e^{\sum_{n=1}^{N} q(z^n=k)\left\langle \log p(v_{1:T}^n, h_{1:T}^n|\Theta^k)\right\rangle_{q(h_{1:T}^n|z^n=k)}},$$

where $\langle\cdot\rangle_q$ denotes expectation with respect to $q$. The specific choice for $p(\Theta^k|\hat{\Theta}^k)$ makes the computation of this posterior relatively straightforward, since $q(\Theta^k)$ is a distribution of the same type. Second, the approximate posterior over the indicator variables is given by

$$q(z^n = k) \propto e^{H_q(h_{1:T}^n|z^n=k)+\langle \log p(z^n=k|z^{\neg n},\gamma)\rangle_{\Pi_{m\neq n} q(z^m)}} e^{\left\langle \log p(v_{1:T}^n,h_{1:T}^n|\Theta^k)\right\rangle_{q(h_{1:T}^n|z^n=k)q(\Theta^k)}},$$

where $H_q(x)$ denotes the entropy of the distribution $q(x)$ and $z^{\neg n}$ includes all indicator variables except for $z^n$. Due to the choice of a Dirichlet prior, the term $p(z^n = k|z^{\neg n}, \gamma) = \int_\pi p(z^n = k|z^{\neg n}, \pi)p(\pi, \gamma)$ can be determined analytically. However, the required average over this term is computationally expensive, and, thus, we approximate it using a second order expansion [13]. The third and most challenging update is the one of the hidden states

$$q\left(h_{1:T}^n|z^n = k\right) \propto e^{\left\langle \log p\left(v_{1:T}^n, h_{1:T}^n|\Theta^k\right)\right\rangle_{q(\Theta^k)}}. \tag{3}$$

Whilst computing this joint density is relatively straightforward, the parameter and indicator variable updates require the non-trivial estimation of the posterior averages $\langle h_t^n \rangle$ and $\langle h_t^n h_{t-1}^n \rangle$ with respect to this distribution. Following a similar approach to the one proposed in [14] for the Bayesian LGSSM, we reformulate the rhs of Equation (3) as proportional to the distribution of an augmented LGSSM such that standard inference routines for the LGSSM can be used.

## 3 Results

In this section we show that the model presented in Section 2 can be used effectively both for inferring the motion templates underlying a set of human trajectories and for approximating motion templates with dynamical systems. For doing so, we take the difficult task of Balero, also known as Ball-In-A-Cup or Kendama, and collect human executions of this task using a motion capture

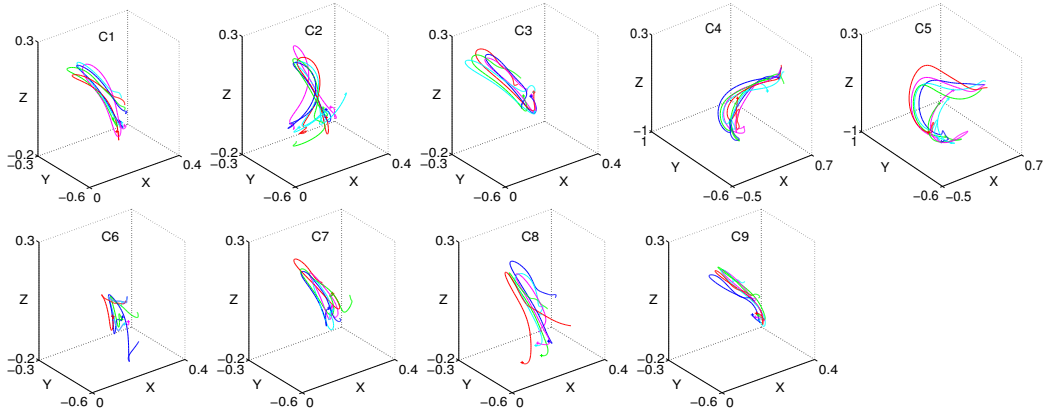

Figure 2: In this figure, we show nine plots where each plot represents one cluster found by our method. Each of the five shown trajectories in the respective clusters represents a different recorded Balero movement. For better visualization, we do not show joint trajectories here but rather the trajectories of the cup which have an easier physical interpretation and, additionally, reveal the differences between the isolated clusters. All axes show units in meters.

setup. We show that the presented model successfully extracts meaningful human motion templates underlying Balero, and that the movements generated by the model are successful in simulation of the Balero task on an anthropomorphic SARCOS arm.

## 3.1 Data Generation of Balero Motions

In the Balero game of dexterity, a human is given a toy consisting of a cup with a ball attached by a string. The goal of the human is to toss the ball into the cup. Humans perform a wide variety of different movements in order to achieve this task [9]. For example, three very distinct movements are: (i) swing the hand slightly upwards to the side and then go back to catch the ball, (ii) hold the cup high and then move very fast to catch the ball, and (iii) jerk the cup upwards and catch the ball in a fast downwards movement. Whilst the difference in these three movements is significant and can be easily detected visually, there exist many other movements for which this is not the case.

We collected 124 different Balero trajectories where the subject was free to select the employed movement. For doing so, we used a VICON$^{TM}$ data collection system which samples the trajectories at 200Hz to track both the cup as well as all seven major degrees of freedom of the human arm. For the evaluation of our method, we considered the seven joint angles of the human presenter as well as the corresponding seven estimated joint velocities.

In the lowest row of Figure 1, we show how the human motion is collected with a VICON$^{TM}$ motion tracking setup. As we will see later, this specific movement is assigned by our method to cluster C2 whose representative generative LGSSM can be used successfully for imitating this motion (middle row). A sketch of the represented movement is shown in the top row of Figure 1.

## 3.2 Clustering and Imitation of Motion Templates

We trained the variational method with different initial conditions, hidden dimension $H = 35$ and a number of clusters $K$ which varied from 20 to 50 in order to avoid suboptimal results due to local maxima.

The resulting clustering contains nine active motion templates. These are plotted in Figure 2, where, instead of the 14-dimensional joint angles and velocities, we show the three-dimensional cup trajectories resulting from these joint movements, as it is easier for humans to make sense of cartesian trajectories. Clusters C1, C2 and C3 are movements to the side which subsequently catch the ball. Here, C1 is a short jerk, C3 appears to have a circular movement similar to a jerky movement, while C2 uses a longer but smoother movement to induce kinetic energy in the ball. Motion templates C4 and C5 are dropping movements where the cup moves down fast for more than 1.2m and then

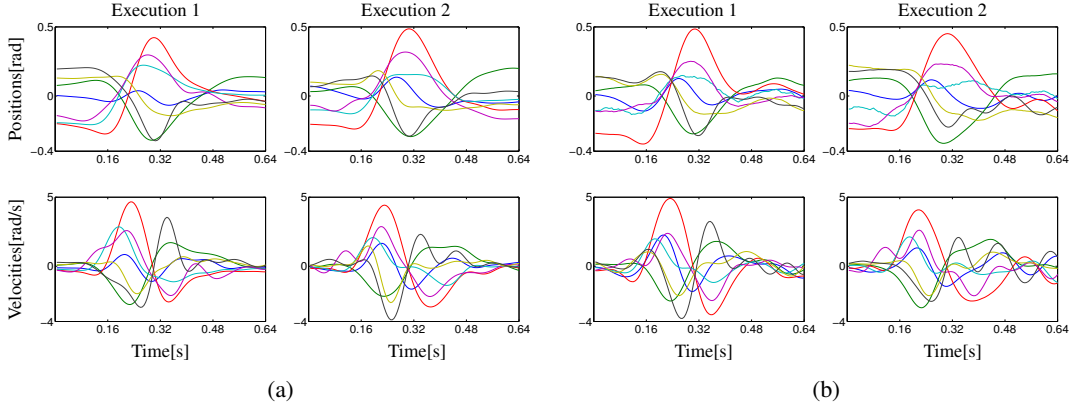

Figure 3: (a) Time-series recorded from two executions of the Balero movement assigned by our model to cluster C1. In the first and second rows are plotted the positions and velocities respectively (for better visualization each time-series component is plotted with its mean removed). (b) Two executions of the Balero movement generated by our trained model using probability distributions of cluster C1.

catches the ball. The template C5 is a smoother movement than C4 with a wider catching movement. For C6 and C7, we observe a significantly different movement where the cup is jerked upwards dragging the ball in this direction and then catches the ball on the way down. Clusters C8 and C9 exhibit the most interesting movement where the main motion is forward-backwards and the ball swings into the cup. In C8 this task is achieved by moving upwards at the same time while in C9 there is little loss of height.

To generate Balero movements with our trained model, we can use the recursive formulation of the LGSSM given by Equation 1 where, for each cluster $k$, $A^k$, $B^k$ and $\mu_1^k$ are replaced by the mean values of their inferred Gaussian $q$ distributions, while the noise covariances are replaced by the modes of their Gamma $q$ distributions. The initial hidden state $h_1$ and the noise elements $\eta_t^h$ and $\eta_t^v$ are sampled from their respective $q$ distributions, whist the inferred optimal values are used for $\mu_{2:T}^k$.

In Figure 3 (a) we plotted two recorded executions of the Balero task assigned by our model to cluster C1. As we can see, the two executions have similar dynamics but also display some differences due to human variability in performing the same type of movement. In Figure 3 (b) we plotted two executions generated by our model using the learned distributions representing cluster C1. Our model can generate time-series with very similar dynamics to the ones of the recorded time-series.

To investigate the accuracy of the obtained motion templates, we used them for executing Balero movements on a simulated anthropomorphic SARCOS arm. Inspired by Miyamoto et al. [15], a small visual feedback term based on a Jacobian transpose method was activated when the ball was within 3cm in order to ensure task-fulfillment. We found that our motion templates are accurate enough to generate successful task executions. This can be seen in Figure 1 for cluster C2 (middle row) and in the video on the author's website.

## 4 Conclusions

In this paper, we addressed the problem of automatic generation of skill libraries for both robot learning and human motion analysis as a unsupervised time-series clustering and learning problem based on human trajectories. We have introduced a novel Bayesian temporal mixture model based on a variational approximation method which is especially designed to enable the use of efficient inference algorithms. We demonstrated that our model gives rise to a meaningful clustering of human executions of the difficult game of dexterity Balero and is able to generate time-series which are very close to the recorded ones. Finally, we have shown that the model can be used to obtain successful executions of the Balero movements on a physically realistic simulation of the SARCOS Master Arm.

## 5 Acknowledgments

The authors would like to thank David Barber for useful discussions and Betty Mohler for help with data collection.

## Footnotes

[1] $v_{1:T}^{1:N}$ is a shorthand for $\left\{v_1^1, \ldots, v_T^1, \ldots, v_1^N, \ldots, v_T^N\right\}$.

[2] Here, $\mathcal{N}(m, S)$ denotes a Gaussian with mean $m$ and covariance $S$, and $0_X$ denotes an $X$-dimensional zero vector. The initial latent state $h_1$ is drawn from $\mathcal{N}(\mu_1, \Sigma)$.

[3] $[X]_{ij}$ and $X_j$ denote the $ij$-th element and the $j$-th column of matrix $X$ respectively. The dependency of the priors on $\Sigma_H$ and $\Sigma_V$ is chosen specifically to render a variational implementation feasible.

[4]Here, we describe a collapsed approximation over $\pi$ [13]. To simplify the notation, we omit conditioning on $v_{1:T}^{1:N}, \hat{\Theta}^{1:K}, \gamma$ for the $q$ distribution.

## References

[1] T. Flash and B. Hochner. Motor primitives in vertebrates and invertebrates. *Current Opinion in Neurobiology*, 15(6):660–666, 2005.

[2] B. Williams, M. Toussaint, and A. Storkey. Modelling motion primitives and their timing in biologically executed movements. In *Advances in Neural Information Processing Systems 20*, pages 1609–1616, 2008.

[3] A. Ijspeert, J. Nakanishi, and S. Schaal. Learning attractor landscapes for learning motor primitives. In *Advances in Neural Information Processing Systems 15*, pages 1547–1554, 2003.

[4] S. Calinon, F. Guenter, and A. Billard. On learning, representing and generalizing a task in a humanoid robot. *IEEE Transactions on Systems, Man and Cybernetics, Part B*, 37(2):286–298, 2007.

[5] J. Durbin and S. J. Koopman. *Time Series Analysis by State Space Methods*. Oxford Univ. Press, 2001.

[6] Y. Xiong and D-Y. Yeung. Mixtures of ARMA models for model-based time series clustering. In *Proceedings of the IEEE International Conference on Data Mining*, pages 717–720, 2002.

[7] C. Li and G. Biswas. A Bayesian approach to temporal data clustering using hidden Markov models. In *Proceedings of the International Conference on Machine Learning*, pages 543–550, 2000.

[8] J. Kober, B. Mohler and J. Peters. Learning perceptual coupling for motor primitives. *International Conference on Intelligent Robots and Systems*, pages 834–839, 2008.

[9] S. Fogel, J. Jacob, and C. Smith. Increased sleep spindle activity following simple motor procedural learning in humans. *Actas de Fisiologia*, 7(123), 2001.

[10] D. J. C. MacKay. *Information Theory, Inference and Learning Algorithms*. Cambridge Univ. Press, 2003.

[11] D. Wipf and J. Palmer and B. Rao. Perspectives on Sparse Bayesian Learning. In *Advances in Neural Information Processing Systems 16*, 2004.

[12] S. Chiappa and D. Barber. Dirichlet Mixtures of Bayesian Linear Gaussian State-Space Models: a Variational Approach. Technical Report no. 161, MPI for Biological Cybernetics, Tübingen, Germany, 2007.

[13] K. Kurihara, M. Welling, and Y. W. Teh. Collapsed variational Dirichlet process mixture models. In *Proceedings of the International Joint Conference on Artificial Intelligence*, pages 2796–2801, 2007.

[14] D. Barber and S. Chiappa. Unified inference for variational Bayesian linear Gaussian state-space models. In *Advances in Neural Information Processing Systems 19*, pages 81–88, 2007.

[15] H. Miyamoto and S. Schaal and F. Gandolfo and Y. Koike and R. Osu and E. Nakano and Y. Wada and M. Kawato. A Kendama learning robot based on bi-directional theory. *Neural Networks*, 9(8): 1281–1302, 1996

